# An Analog VLSI Model of the Fly Elementary Motion Detector

**Reid R. Harrison and Christof Koch**
Computation and Neural Systems Program, 139-74
California Institute of Technology
Pasadena, CA 91125
[harrison,koch]@klab.caltech.edu

## Abstract

Flies are capable of rapidly detecting and integrating visual motion information in behaviorly-relevant ways. The first stage of visual motion processing in flies is a retinotopic array of functional units known as elementary motion detectors (EMDs). Several decades ago, Reichardt and colleagues developed a correlation-based model of motion detection that described the behavior of these neural circuits. We have implemented a variant of this model in a 2.0-$\mu m$ analog CMOS VLSI process. The result is a low-power, continuous-time analog circuit with integrated photoreceptors that responds to motion in real time. The responses of the circuit to drifting sinusoidal gratings qualitatively resemble the temporal frequency response, spatial frequency response, and direction selectivity of motion-sensitive neurons observed in insects. In addition to its possible engineering applications, the circuit could potentially be used as a building block for constructing hardware models of higher-level insect motion integration.

## 1 INTRODUCTION

Flies rely heavily on visual motion information to survive. In the fly, motion information is known to underlie many important behaviors including stabilization during flight, orienting towards small, rapidly-moving objects (Egelhaaf and Borst 1993), and estimating time-to-contact for safe landings (Borst and Bahde 1988). Some motion-related tasks like extending the legs for landing can be excecuted less than 70 milliseconds after stimulus presentation. The computational machinery performing this sensory processing is fast, small, low-power, and robust.

There is good evidence that motion information is first extracted by local elementary motion detectors (see Egelhaaf *et al.* 1988 and references therein). These EMDs are ar-

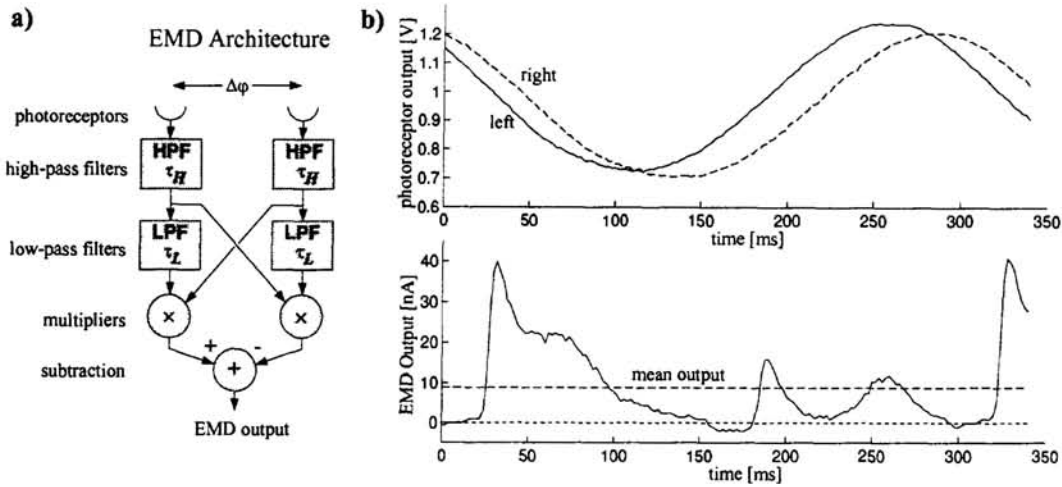

Figure 1: Elementary Motion Detector. a) A simplified version of our EMD circuit architecture. In the actual circuit implementation, there are separate ON and OFF channels that operate in parallel. These two channels are summed after the muliplication. b) The measured response of the EMD test circuit to a drifting sinusoidal grating. Notice that the output is phase dependent, but has a positive mean response. If the grating was drifting in the opposite direction, the circuit would give a negative mean response.

ranged retinotopically, and receive input from adjacent photoreceptors. The properties of these motion-sensitive units have been studied extensively during the past 30 years. Direct recording from individual EMDs is difficult due to the small size of the cells, but much work has been done recording from large tangential cells that integrate the outputs of many EMDs over large portions of the visual field. From these studies, the behavior of individual EMDs has been inferred.

If we wish to study models of motion integration in the fly, we first need a model of the EMD. Since many motion integration neurons in the fly are only a few synapses away from muscles, it may be possible in the near future to contruct models that complete the sensorimotor loop. If we wish to include the real world in the loop, we need a mobile system that works in real time. In the pursuit of such a system, we follow the neuromorphic engineering approach pioneered by Mead (Mead 1989) and implement a hardware model of the fly EMD in a commercially available 2.0-$\mu m$ CMOS VLSI process. All data presented in this paper are from one such chip.

## 2  ALGORITHM AND ARCHITECTURE

Figure 1a shows a simplified version of the motion detector. This is an elaborated version of the correlation-based motion detector first proposed by Reichardt and colleagues (see Reichardt 1987 and references therein). The Reichardt motion detector works by correlating (by means of a multiplication) the response of one photoreceptor to the delayed response of an adjacent photoreceptor. Our model uses the phase lag inherent in a low-pass filter to supply the delay. The outputs from two mirror-symmetric correlators are subtracted to remove any response to full-field flicker ($\omega_s = 0, \omega_t > 0$).

Correlation-based EMDs are not pure velocity sensors. Their response is strongly affected by the contrast and the spatial frequency components of the stimulating pattern. They can best be described as direction-selective spatiotemporal filters. The mean steady-state response $R$ of the motion detector shown in Figure 1a to a sinusoidal grating drifting in one direction can be expressed as a separable function of stimulus amplitude ($\Delta I$), temporal

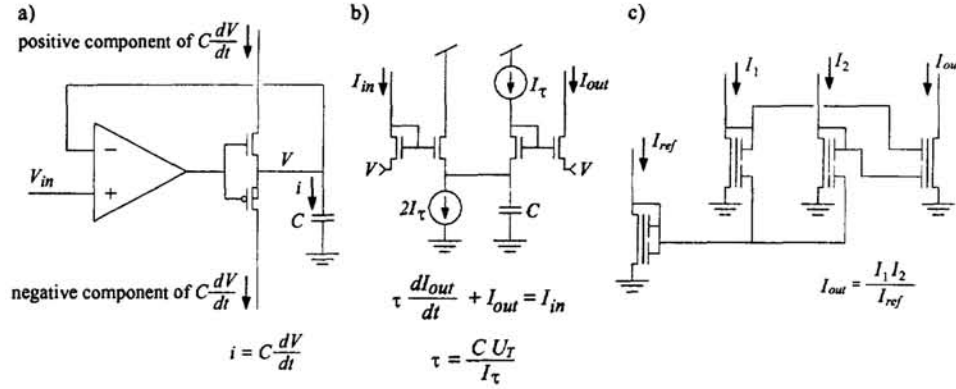

$$i = C\frac{dV}{dt}$$

$$\tau\frac{dI_{out}}{dt} + I_{out} = I_{in}$$

$$\tau = \frac{C\,U_T}{I_\tau}$$

$$I_{out} = \frac{I_1 I_2}{I_{ref}}$$

Figure 2: EMD Subcircuits. a) Temporal derivative circuit. In combination with the first-order low-pass filter inherent in the photoreceptor, this forms the high-pass filter with time constant $\tau_H$. The feedback amplifier enforces $V = V_{in}$, and the output is the current needed for the nFET or pFET source follower to charge or discharge the capacitor $C$. b) Current-mode low-pass filter. The time constant $\tau_L$ is determined by the bias current $I_\tau$ (which is set by a bias voltage supplied from off-chip), the capacitance $C$, and the thermal voltage $U_T = kT/q$. c) Current-mode one-quadrant multiplier. The devices shown are floating-gate nFETs. Two control gates capacitively couple to the floating node, forming a capacitive divider.

frequency ($\omega_t = 2\pi f_t$), and spatial frequency ($\omega_s = 2\pi f_s$):

$$
\begin{aligned}
R(\Delta I, \omega_t, \omega_s) &= R_I(\Delta I) \times R_t(\omega_t) \times R_s(\omega_s) & (1)\\
&= \Delta I^2 \times \frac{\tau_L \omega_t}{(1 + 1/\tau_H^2\omega_t^2)(1 + \tau_L^2\omega_t^2)} \times \sin(\Delta\varphi\omega_s) & (2)
\end{aligned}
$$

where $\Delta\varphi$ is the angular separation of the photoreceptors, $\tau_H$ is the time constant of the high-pass filter, and $\tau_L$ is the time constant of the low-pass filter (see Figure 1a). (Note that this holds only for motion in a particular direction. Motion detectors are not linearly separable overall, but the single-direction analysis is useful for making comparisons.)

## 3   CIRCUIT DESCRIPTION

In addition to the basic Reichardt model described above, we include a high-pass filter in series with the photoreceptor. This amplifies transient responses and removes the DC component of the photoreceptor signal. We primarily use the high-pass filter as a convenient circuit to switch from a voltage-mode to a current-mode representation (see Figure 2a).

For the photoreceptor, we use an adaptive circuit developed by Delbrück (Delbrück and Mead 1996) that produces an output voltage proportional to log intensity. We bias the photoreceptor very weakly to attenuate high temporal frequencies. This is directly followed by a temporal derivative circuit (Mead 1989) (see Figure 2a), the result being a high-pass filter with the dominant pole $\tau_H$ being set by the photoreceptor cutoff frequency. The outputs of the temporal derivative circuit are two unidirectional currents that represent the positive and negative components of a high-pass filtered version of the photoreceptor output. This resembles the ON and OFF channels found in many biological visual systems. Some studies suggest ON and OFF channels are present in the fly (Franceschini *et al.* 1989) but the evidence is mixed (Egelhaaf and Borst 1992). This two-channel representation is useful for current-mode circuits, since the following translinear circuits work only with unidirectional

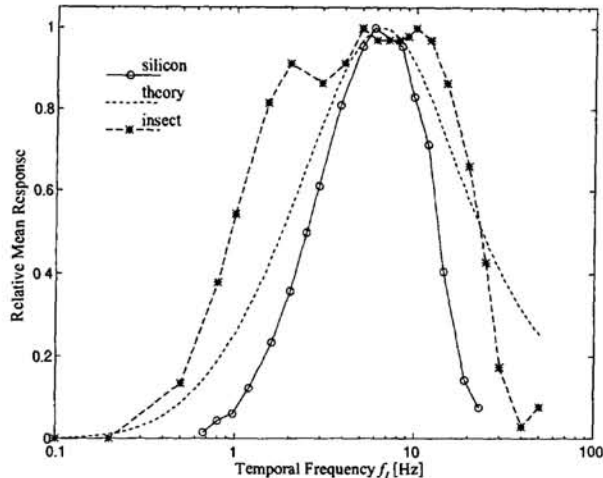

Figure 3: Temporal Frequency Response. Circuit data was taken with $f_s$ = 0.05 cycles/deg and 86% contrast. Theory trace is $R_t(\omega_t)$ from Equation 2, where $\tau_H$ = 360 *ms* and $\tau_L$ = 25 *ms* were directly measured in separate experiments – these terms were not fit to the data. Insect data was taken from a wide-field motion neuron in the blowfly *Calliphora erythrocephala* (O'Carroll *et al.* 1996). All three curves were normalized by their peak response.

currents. It should be noted that the use of ON and OFF channels introduces nonlinearities into the circuit that are not accounted for in the simple model described by Equation 2.

The current-mode low-pass filter is shown in Figure 2b. The time constant $\tau_L$ is set by the bias current $I_\tau$. This is a log-domain filter that takes advantage of the exponential behavior of field-effect transistors (FETs) in the subthreshold region of operation (Minch, personal communication).

The current-mode multiplier is shown in Figure 2c. This circuit is also translinear, using a diode-connected FET to convert the input currents into log-encoded voltages. A weighted sum of the voltages is computed with a capacitive divider, and the resulting voltage is exponentiated by the output FET into the output current. The capacitive divider creates a floating node, and the charge on all these nodes must be equalized to ensure matching across independent multipliers. This is easily accomplished by exposing the chip to UV light for several minutes. This circuit represents one of a family of floating-gate MOS translinear circuits developed by Minch that are capable of computing arbitrary power laws in current mode (Minch *et al.* 1996).

After the multiplication stage, the currents from the ON and OFF channels are summed, and the final subtraction of the left and right channels is done off-chip. There is a gain mismatch of approximately 2.5 between the left and right channels that is now compensated for manually. This mismatch must be lowered before large on-chip arrays of EMDs are practical. A new circuit designed to lessen this gain mismatch is currently being tested. It is interesting to note that there is no significant offset error in the output currents from each channel. This is a consequence of using translinear circuits which typically have gain errors due to transistor mismatch, but no fixed offset errors.

## 4 EXPERIMENTS

As we showed in Equation 2, the motion detector's response to a drifting sinusoidal grating of a particular direction should be a separable function of $\Delta I$, temporal frequency, and

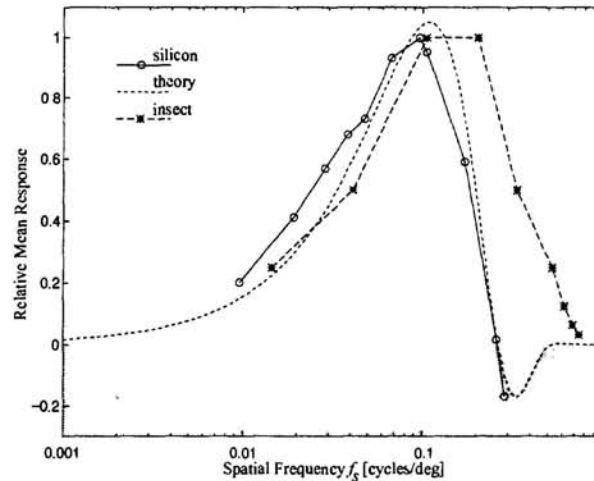

Figure 4: Spatial Frequency Response. Circuit data was taken with $f_t = 4$ Hz and 86% contrast. Theory trace is $R_s(\omega_s)$ from Equation 2 multiplied by $\exp(-\omega_s{}^2/K^2)$ to account for blurring in the optics. The photoreceptor spacing $\Delta\varphi = 1.9°$ was directly measured in an separate experiment. Only $K$ and the overall magnitude were varied to fit the data. Insect data was taken from a wide-field motion neuron in the hoverfly *Volucella pelluscens* (O'Carroll *et al.* 1996). Circuit and insect data were normalized by their peak response.

spatial frequency. We tested the circuit along these axes using printed sinusoidal gratings mounted on a rotating drum. A lens with an 8-*mm* focal length was mounted over the chip. Each stimulus pattern had a fixed contrast $\Delta I/2\overline{I}$ and spatial frequency $f_s$. The temporal frequency was set by the pattern's angular velocity $v$ as seen by the chip, where $f_t = f_s v$.

The response of the circuit to a drifting sine wave grating is phase dependent (see Figure 1b). In flies, this phase dependency is removed by integrating over large numbers of EMDs (spatial integration). In order to evaluate the performance of our circuit, we measured the mean response over time.

Figure 3 shows the temporal frequency response of the circuit as compared to theory, and to a wide-field motion neuron in the fly. The circuit exhibits temporal frequency tuning. The point of peak response is largely determined by $\tau_L$, and can be changed by altering the low-pass filter bias current. The deviation of the circuit behavior from theory at low frequencies is thought to be a consequence of crossover distortion in the temporal derivative circuit. At high temporal frequencies, parasitic capacitances in current mirrors are a likely candidate for the discrepancy. The temporal frequency response of the blowfly *Calliphora* is broader than both the theory and circuit curves. This might be a result of time-constant adaptation found in blowfly motion-sensitive neurons (de Ruyter van Steveninck *et al.* 1986).

Figure 4 shows the spatial frequency response of the circuit. The response goes toward zero as $\omega_s$ approaches zero, indicating that the circuit greatly attenuates full-field flicker. The circuit begins aliasing at $\omega_s = 1/2\Delta\varphi$, giving a response in the wrong direction. Spatial aliasing has also been observed in flies (Götz 1965). The optics used in the experiment act as an antialiasing filter, so aliasing could be avoided by defocusing the lens slightly.

Figure 5 shows the directional tuning of the circuit. It can be shown that as long as the spatial wavelength is large compared to $\Delta\varphi$, the directional sensitivity of a correlation-based motion detector should approximate a cosine function (Zanker 1990). The circuit's performance matches this quite well. Motion sensitive neurons in the fly show cosine-like direction selectivity.

Figure 6 shows the contrast response of the circuit. Insect EMDs show a saturating contrast

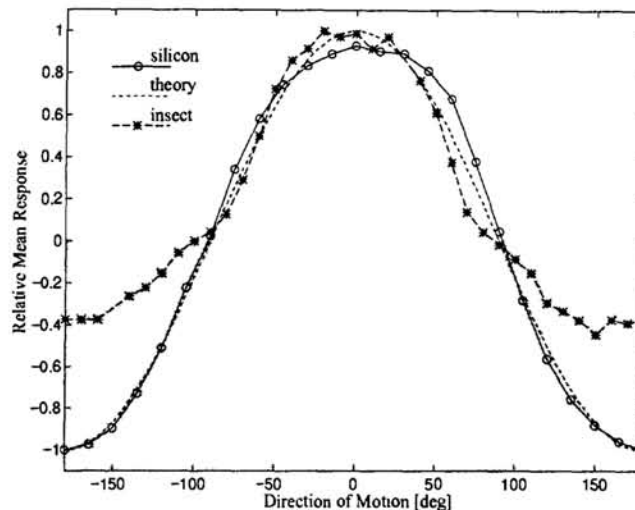

Figure 5: Directional Response. Circuit data was taken with $f_t = 6$ Hz, $f_s = 0.05$ cycles/deg and 86% contrast. Theory trace is $\cos\alpha$, where $\alpha$ is the direction of motion relative to the axis along the two photoreceptors. Insect data was taken from the H1 neuron in the blowfly *Calliphora erythrocephala* (van Hateren 1990). H1 is a spiking neuron with a low spontaneous firing rate. The flattened negative responses visible in the graph are a result of the cell's limited dynamic range in this region. All three curves were normalized by their peak response.

response curve, which can be accounted for by introducing saturating nonlinearities before the multiplication stage (Egelhaaf and Borst 1989). We did not attempt to model contrast saturation in our circuit, though it could be added in future versions.

## 5 CONCLUSIONS

We implemented and tested an analog VLSI model of the fly elementary motion detector. The circuit's spatiotemporal frequency response and directional selectivity is qualitatively similar to the responses of motion-sensitive neurons in the fly. This circuit could be a useful building block for constructing analog VLSI models of motion integration in flies. As an integrated, low-power, real-time sensory processor, the circuit may also have engineering applications.

### Acknowledgements

This work was supported by the Center for Neuromorphic Systems Engineering as a part of NSF's Engineering Research Center program, and by ONR. Reid Harrison is supported by an NDSEG fellowship from ONR. We thank Bradley Minch, Holger Krapp, and Rainer Deutschmann for invaluable discussions.

## References

A. Borst and S. Bahde (1988) Visual information processing in the fly's landing system. *J. Comp. Physiol. A* **163:** 167-173.

T. Delbrück and C. Mead (1996) Analog VLSI phototransduction by continuous-time, adaptive, logarithmic photoreceptor circuits. *CNS Memo No. 30,* Caltech.

M. Egelhaaf, K. Hausen, W. Reichardt, and C. Wehrhahn (1988) Visual course control in

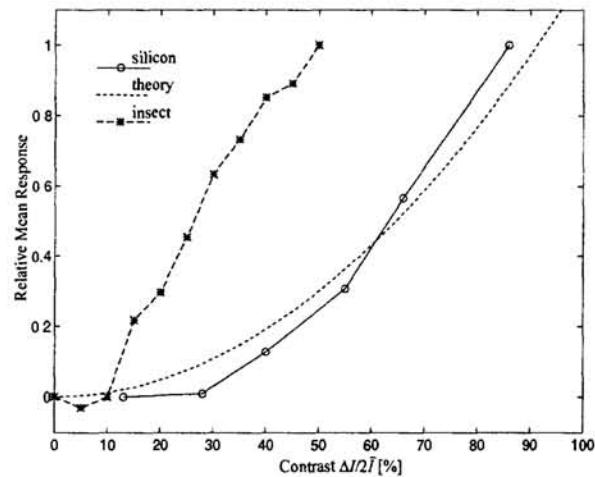

Figure 6: Contrast Response. Circuit data was taken with $f_t = 6$ Hz and $f_s = 0.1$ cycles/deg. Theory trace is $R_I(\Delta I)$ from Equation 2 with its magnitude scaled to fit the circuit data. Insect data was taken from the HS neuron in the blowfly *Calliphora erythrocephala* (Egelhaaf and Borst 1989). Circuit and insect data were normalized by their peak response.

flies relies on neuronal computation of object and background motion. *TINS* **11**: 351-358.

M. Egelhaaf and A. Borst (1989) Transient and steady-state response properties of movement detectors. *J. Opt. Soc. Am. A* **6**: 116-127.

M. Egelhaaf and A. Borst (1992) Are there separate ON and OFF channels in fly motion vision? *Visual Neuroscience* **8**: 151-164.

M. Egelhaaf and A. Borst (1993) A look into the cockpit of the fly: Visual orientation, algorithms, and identified neurons. *J. Neurosci.* **13**: 4563-4574.

N. Franceschini, A. Riehle, and A. le Nestour (1989) Directionally selective motion detection by insect neurons. In Stavenga/Hardie (eds.), *Facets of Vision*, Berlin: Springer-Verlag.

K.G. Götz (1965) Die optischen Übertragungseigenschaften der Komplexaugen von *Drosophila. Kybernetik* **2**: 215-221.

J.H. van Hateren (1990) Directional tuning curves, elementary movement detectors, and the estimation of the direction of visual movement. *Vision Res.* **30**: 603-614.

C. Mead (1989) *Analog VLSI and Neural Systems*. Reading, Mass.: Addison-Wesley.

B.A. Minch, C. Diorio, P. Hasler, and C. Mead (1996) Translinear circuits using subthreshold floating-gate MOS transistors. *Analog Int. Circuits and Signal Processing* **9**: 167-179.

D.C. O'Carroll, N.J. Bidwell, S.B. Laughlin, and E.J. Warrant (1996) Insect motion detectors matched to visual ecology. *Nature* **382**: 63-66.

W. Reichardt (1987) Evaluation of optical motion information by movement detectors. *J. Comp. Phys. A* **161**: 533-547.

R.R. de Ruyter van Steveninck, W.H. Zaagman, and H.A.K. Mastebroek (1986) Adaptation of transient responses of a movement-sensitive neuron in the visual system of the blowfly *Calliphora erythrocephala. Biol. Cybern.* **54**: 223-236.

J.M. Zanker (1990) On the directional sensitivity of motion detectors. *Biol. Cybern.* **62**: 177-183.
